# FEEL-SNN: Robust Spiking Neural Networks with Frequency Encoding and Evolutionary Leak Factor

**Mengting Xu**[1,2]   **De Ma** [1,2]   **HuaJin Tang** [1,2]   **Qian Zheng**[1,2] *   **Gang Pan** [1,2] *

[1] The State Key Lab of Brain-Machine Intelligence, Zhejiang University, Hangzhou, China
[2] College of Computer Science and Technology, Zhejiang University, Hangzhou, China
{xumengting, made, htang, qianzheng, gpan}@zju.edu.cn

## Abstract

Currently, researchers think that the inherent robustness of spiking neural networks (SNNs) stems from their biologically plausible spiking neurons, and are dedicated to developing more bio-inspired models to defend attacks. However, most work relies solely on experimental analysis and lacks theoretical support, and the direct-encoding method and fixed membrane potential leak factor they used in spiking neurons are simplified simulations of those in the biological nervous system, which makes it difficult to ensure generalizability across all datasets and networks. Contrarily, the biological nervous system can stay reliable even in a highly complex noise environment, one of the reasons is selective visual attention and non-fixed membrane potential leaks in biological neurons. This biological finding has inspired us to design a highly robust SNN model that closely mimics the biological nervous system. In our study, we first present a unified theoretical framework for SNN robustness constraint, which suggests that improving the encoding method and evolution of the membrane potential leak factor in spiking neurons can improve SNN robustness. Subsequently, we propose a robust SNN (FEEL-SNN) with Frequency Encoding (FE) and Evolutionary Leak factor (EL) to defend against different noises, mimicking the selective visual attention mechanism and non-fixed leak observed in biological systems. Experimental results confirm the efficacy of both our FE, EL, and FEEL methods, either in isolation or in conjunction with established robust enhancement algorithms, for enhancing the robustness of SNNs. Our code is available at https://github.com/zju-bmi-lab/FEEL_SNN.

## 1   Introduction

In recent years, brain-inspired spiking neural networks (SNNs) [21] have been increasingly prominent. Unlike traditional artificial neural networks (ANNs), which process a single image using floating-point values, spiking neural networks encode spatial-pixel image into temporal spike train. Information is transmitted by the occurrence of spikes (using 0 to signify no spike and 1 to denote a spike) whenever the membrane potential of a spiking neuron exceeds its threshold, thereby emulating biological neurons [30, 37, 14]. The distinctive spatio-temporal characteristics, discrete representation, and event-driven properties of SNNs enable them to operate efficiently on neuromorphic hardware [26, 4, 20, 46]. This makes them increasingly applicable to a variety of tasks [42, 19, 48], such as spatio-temporal pattern recognition [41] and high-speed detection [15]. As SNNs attract increasing attention from academia and industry, the issue of security [35] becomes more important. When SNNs are applied to safety-critical systems, their reliability should be a major concern [10]. While SNNs have demonstrated better robustness compared to ANNs [11, 32, 31], recent studies have shown that they are still vulnerable to noise [17, 8]. Among all types of perturbations, adversarial noise [35], which

---

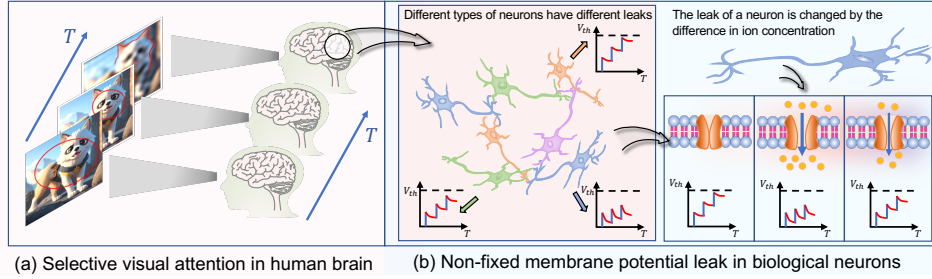

Figure 1: Illustration of the (a) selective visual attention and (b) non-fixed membrane potential leak in biological nervous system.

refers to visually imperceptible alterations that can mislead a well-trained network, is particularly concerning. Therefore, improving the robustness of SNNs is crucial for their real-life deployment.

At present, researchers believe that the inherent robustness of SNN is brought by its more biologically plausible spiking neurons, and they are dedicated to developing more bio-inspired models to cope with noise attacks. Among them, some work focuses on SNN robustness analysis. [11] investigates the SNNs robustness to adversarial attacks with different values of the neuron's firing voltage thresholds and time window boundaries. [9] suggests that the utility of spike timing in SNNs could improve the robustness against attacks. [39, 32, 3] analyze the adversarial accuracy of SNNs trained with leak factor in LIF spiking neurons. However, these analyses rely solely on experimental analysis and lack sufficient theoretical support to ensure generalizability across all datasets and networks. Other work aims to improve the robustness of SNN from biological aspects. [10] further introduces stochasticity in biological neurons as a stochastic gating mechanism for spiking neurons to enhance model robustness. However, this may result in the loss of the original information. The noise environment encountered by the biological nervous system is highly complex, with various types of noise spanning different frequency ranges [43]. Adversarial noises also exhibit different frequencies across datasets, rather than being fixed [23, 1]. Despite this complexity, the biological nervous system maintains robustness. This inspired us to design a highly robust SNN model that more closely mimics the biological nervous system, allowing it to adapt to and overcome the challenges posed by diverse and dynamic noise environments.

For the biological brain, as shown in Fig. (1a), there exists the selective visual attention mechanism that selectively focuses on stimuli of different frequencies over time and can filter out unwanted information [7, 45]. This aids the biological nervous system in avoiding the instability caused by noise [44]. Additionally, as shown in Fig. (1b), the changes in membrane potential in biological neurons are determined by ion concentration inside and outside the cell membrane. Different environments and types of nerve axon fibers can affect the degree of leak of the membrane potential [38, 13], which contributes to the biological nervous system's processing of complex noise [33]. Motivated by these biological insights, we undertake a theoretical examination of the robustness of SNNs. We present a unified framework for SNN robustness constraint, which indicates that refining the encoding technique and evolution of the membrane potential leak factor can enhance SNN robustness. Subsequently, we propose a robust SNN with Frequency Encoding and Evolutionary Leak factor (FEEL-SNN) to defend against different noises. Our main contributions are summarized as follows:

- Through analysis of the model's adversarial loss, we theoretically present a unified framework for SNN robustness. Our findings suggest that enhancing the encoding method and evolution of the membrane potential leak factor can improve SNN robustness.

- We propose a frequency encoding (FE) method for SNNs. FE captures information of varying frequencies at different time steps, mimicking the selective visual attention mechanism observed in biological systems. FE can preserve the original information while suppressing different frequency range noises, effectively filtering out image noise.

- Based on FE, we propose an evolutionary membrane potential leak factor (EL). EL ensures that different neurons in the network learn the optimal robustness leak factor at different time steps, which is aimed at learning the correlation between frequencies at different time steps. It simulates the membrane potential leak in biological neurons and ensures an enhancement in model robustness.

- Experimental results validate that both our FE and EL methods can effectively improve the robustness of SNN to different noises, and can be used in conjunction with other methods to improve the robustness further.

## 2   Related Work

[24, 31] demonstrate that compared to ANNs, SNNs exhibit inherent robustness. Currently, researchers consider that this superior robustness of SNNs stems from their more biologically plausible spiking neurons, and they are dedicated to developing more bio-inspired models to cope with noise attacks. [11] investigate the security of SNNs from the impact of structural parameters on the robustness of SNNs to adversarial attacks, and demonstrate that the inherent robustness of SNNs is highly conditioned by the choice of (time window; firing voltage thresholds) combination. [9] systematically demonstrates that precise spike timing is conducive to improving the robustness of neural networks, providing opportunities for understanding the robustness of the brain. [32] confirms that the leak factor in LIF neurons offers an extra knob to control the adversarial perturbation. [3] also investigates the role played by leak factor and concludes from frequency domain analysis that leak factor can filter high-frequency components thus improving model robustness. However, the spiking neurons used in these works only offer a simplified representation of the intricate dynamics of the biological system [10], and the robustness verification of the above works is mainly carried out experimentally, lacking a theoretical explanation. It's doubtful that their conclusion can adapt to other datasets and varying noises. Then, StoG [10] is proposed to further introduce stochasticity which is observed in biological neurons into the spiking neurons. The more biologically plausible StoG method can improve the robustness efficiently, however, it sacrifices a little original accuracy. In contrast to the aforementioned work, we conducted a theoretical analysis of the robustness of SNNs, showing that it is constrained by the encoding method and the membrane potential leak factor. We then designed a frequency encoding and evolutionary leak factor model that closely mimics the biological nervous system to enhance the robustness of SNNs.

Another method improves the robustness of SNNs by incorporating additional training strategies. [17] continuously adds newly generated adversarial examples during the training process to improve the robustness of SNN. [8] proposes a regularized adversarial training scheme by performing the Lipschitz analysis on model weights. However, these methods are rooted in the concept of adversarial training [22] in ANNs, and their defense performance can be influenced by downstream tasks. Furthermore, they all rely on a simplified direct encoding approach (i.e., repeating the original image $T$ times) as input, which deviates from the visual attention mechanism observed in biological brains. Developing more biologically plausible models is essential for improving robustness and advancing SNN applications. In this work, we leverage the selective visual attention mechanism found in biological brains and introduce a frequency encoding method. This method effectively filters noise in images, enhancing the robustness of SNNs.

## 3   Preliminaries

### 3.1   Spiking Neurons

The most commonly used spiking neuron of SNNs today is the Leaky Integrate-and-Fire (LIF) spiking neuron [36, 6]. LIF neurons simplify and computationally simulate the three main processes involved in information transmission in biological neurons: synaptic integration, membrane potential accumulation and decay, and neuronal firing. The dynamics of LIF spiking neurons in layer $l$ can be described as follows:

$$Synaptic \quad integration: \quad \boldsymbol{m}_l^t = \begin{cases} \boldsymbol{W}_{l-1,l}\boldsymbol{O}_{l-1}^t, & l > 1 \\ \boldsymbol{x}^t. & l = 1 \end{cases} \tag{1}$$

$$Membrane \quad potential \quad accumulation: \quad \boldsymbol{u}_l^t = \boldsymbol{\lambda}_l^t \boldsymbol{u}_l^{t-1} \odot (1 - \boldsymbol{O}_l^{t-1}) + \boldsymbol{m}_l^t. \tag{2}$$

$$Neuronal \quad firing: \quad \boldsymbol{O}_l^t = \mathcal{H}(\boldsymbol{u}_l^t - V_{th}). \tag{3}$$

In the first layer, the injected electrical signal $\boldsymbol{m}_1^t$ accumulates from the input signal $\boldsymbol{x}^t$. For subsequent layers $l > 1$, the electrical signal $\boldsymbol{m}_l^t$ is the sum of spike signals $\boldsymbol{O}_{l-1}^t$ from the preceding

layer scaled by weights $\boldsymbol{W}_{l-1,l}$. The operator $\odot$ represents element-wise product. The membrane potential $\boldsymbol{u}_l^t$ at time step $t$ is the accumulation of the leaked membrane potential $\boldsymbol{u}_l^{t-1}$ from the previous time step and the newly injected signal $\boldsymbol{m}_l^t$. If the membrane potential $\boldsymbol{u}_l^t$ exceeds the threshold $V_{th}$, a spike $\boldsymbol{O}_l^t$ is generated by a Heaviside function $\mathcal{H}(\cdot)$. After spikes are produced, the membrane potential of the corresponding neurons is reset to 0. Typically, $\boldsymbol{\lambda}_l^t$ is treated as a constant value $\lambda \in (0, 1]$ in previous work.

## 3.2 Adversarial Attacks

Given a classification model $f$ with dataset $(\boldsymbol{x}, y_{true})$, where $\boldsymbol{x}$ is the clean image and $y_{true}$ is the corresponding label. The adversarial attack aims to generate an adversarial example $\hat{\boldsymbol{x}}$ that satisfies:

$$f(\hat{\boldsymbol{x}}) \neq f(\boldsymbol{x}) \quad s.t. \quad ||\hat{\boldsymbol{x}} - \boldsymbol{x}||_p \leq \epsilon, \tag{4}$$

where$||\cdot||_p$ is the $L_p$-norm, we use $L_\infty$-norm on our work, and $\epsilon$ limits the strength of the perturbation to a level that is indistinguishable to the human eye. Here we consider four classic adversarial attack algorithms: Fast Gradient Sign Method (FGSM) [12], Projected Gradient Descent (PGD) [22], Basic iterative Method (BIM) [18] and CW [2] attacks. The detailed formulations of these attacks can be found in the Appendix A.1. The introduction of surrogate functions [25, 47, 41] addresses the limitation of backpropagating gradients through LIF neurons. This advancement enables effective adversarial attacks on SNNs using the aforementioned methods.

# 4 FEEL-SNN: Robust SNNs with Frequency Encoding and Evolutionary Leak Factor

## 4.1 The robustness analysis of SNNs

The robustness of the model is quantified as $\mathcal{L}(\boldsymbol{x} + \epsilon) - \mathcal{L}(\boldsymbol{x})$, the difference in loss value before and after perturbation. Improving robustness entails reducing this perturbation-induced loss difference. [27] utilizes the local linearity technique to theoretically address this difference, expressed as:

$$\mathcal{L}(\boldsymbol{x} + \epsilon) - \mathcal{L}(\boldsymbol{x}) \leq |\epsilon \odot \nabla_{\boldsymbol{x}} \mathcal{L}(x)|_1 + g(\epsilon, \boldsymbol{x}), \tag{5}$$

where $g(\epsilon, \boldsymbol{x})$ is the residual term, $|\cdot|_1$ is $l_1$ norm for vector. This theoretical framework motivates research into regularization that minimize $|\epsilon \odot \nabla_{\boldsymbol{x}} \mathcal{L}(x)|_1$ in ANNs to enhance robustness [18, 29].

The situation for SNNs differs slightly from that for ANNs [10]. In SNNs, the perturbed input $\hat{\boldsymbol{x}} = \boldsymbol{x} + \epsilon$ is encoded into temporal trains over $T$ time steps. Consequently, the robustness constraint for SNNs should aim to minimize the term $\sum_t |\epsilon(t) \odot \frac{\partial \mathcal{L}}{\partial \boldsymbol{x}^t}|_1$ according to Eq. (5), where $\boldsymbol{x}^t$ is the input encoding image at time step $t$, and $\epsilon(t)$ represents the perturbation of the encoding image $\boldsymbol{x}^t$ at time step $t$. By applying the BPTT rule [41], we can derive the constraint for the term $\sum_t |\epsilon(t) \odot \frac{\partial \mathcal{L}}{\partial \boldsymbol{x}^t}|_1$ in SNNs, as presented in Theorem 1 (The detailed proof is in the Appendix A.2).

**Theorem 1** *Given an L-layered SNN intended to inference $T$ time-steps with $\boldsymbol{\lambda}$ as the leak factor, suppose that there are $N_l$ neurons in layer $l$ for $l = 1, 2, \ldots L$. $\boldsymbol{\lambda}_l \in \mathbb{R}^{N_l \times T}$, $\boldsymbol{W}_{l-1,l} \in \mathbb{R}^{N_l \times N_{l-1}}$, it satisfies:*

$$\min \sum_t |\epsilon(t) \odot \frac{\partial \mathcal{L}}{\partial \boldsymbol{x}^t}|_1 = \min \sum_t |\frac{1}{L}\sum_{l=1}^{L}[(\underbrace{\prod_{k=t}^{T} \epsilon(t) \odot \boldsymbol{\lambda}_l^k}_{①}) \cdot \underbrace{\prod_{q=2}^{l} \boldsymbol{W}_{q-1,q}}_{②} \cdot \underbrace{\prod_{v=1}^{l} \frac{\partial \boldsymbol{O}_v^t}{\partial \boldsymbol{u}_v^t}}_{③} \cdot \frac{\partial \mathcal{L}}{\partial \boldsymbol{O}_l^T}]|_1, \quad (6)$$

*where $\epsilon$ is the perturbation, $\mathcal{L}$ is the loss function.*

According to Eq. (6), the robustness of SNNs is relative to the perturbation $\epsilon$ and the leak factor $\boldsymbol{\lambda}$ in the ① term, the model weight $\boldsymbol{W}$ in the ② term, and the $\frac{\partial \boldsymbol{O}_v^t}{\partial \boldsymbol{u}_v^t}$ in the ③ term. Eq. (6) presents a unified framework for SNNs robustness constraint, which helps explain why weight regularization [8] (the ② term) and surrogate gradient [40] (the ③ term ) can promote robustness. And the previous work [39, 32, 3] also analyzes the inherent robustness of SNNs from the leak factor in the ① term. However, there is still a lack of work on removing input perturbation and improving leak factors in the ① term to enhance the robustness of SNNs.

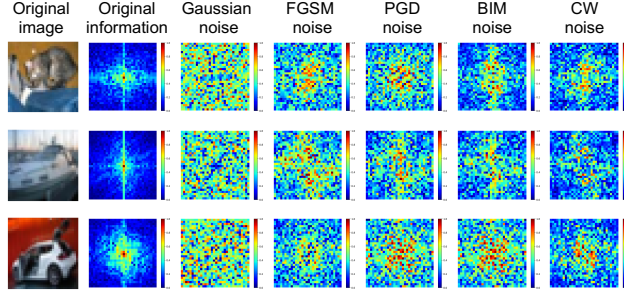

Figure 2: Visualization frequency spectrums for data observation. The first column shows three cases of original CIFAR10 images. The second column shows the corresponding frequency spectrums of the images in the first column. The third column to the seventh column shows the frequency spectrums of corresponding added noises to the images in the first column, where added noise maps the difference between the noise image and the original one. The center of each frequency spectrum represents the low-frequency information, and the edge area is the high-frequency information.

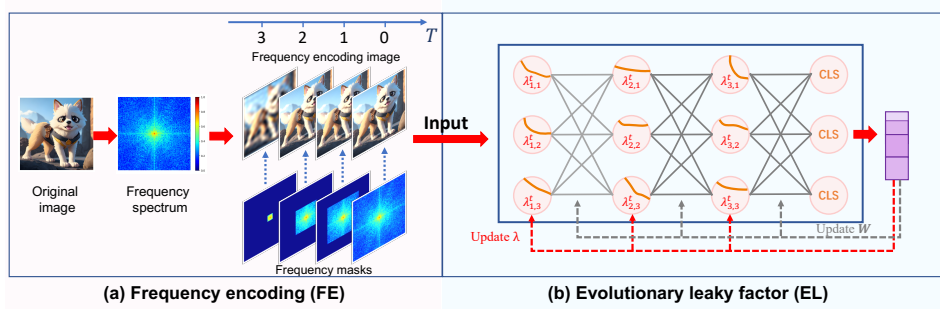

Figure 3: Illustration of the proposed FEEL-SNN. (a) Frequency encoding to simulate the selective visual attention in biological brain and (b) Evolutionary leak factor to simulate the non-fixed membrane potential leak in biological nervous system.

### 4.2 Frequency encoding to simulate the selective visual attention in biological brain

According to the ① term in Eq. (6), reducing input perturbations $\epsilon(t)$ at every time step helps the model achieve reliable output. However, the encoding method that much of the current work relies on is the simplified direct encoding approach [11, 9, 10] (*i.e.*, repeating $T$ images), which repeats the noise $T$ times and inevitably overlooks noise removal. In contrast, the biological nervous system can maintain stability even in complex noise environments, benefiting from the selective visual attention mechanism [7, 45] of the brain (as illustrated in Fig. (1a). The brain processes only a fraction of the information available on the retina at any given time and has the ability to filter out unwanted information. To simulate the selective visual attention of the biological nervous system while effectively removing noise at different frequencies, we propose the Frequency Encoding method (FE) for SNNs. Specifically, given an input image, FE transforms it into the frequency domain via Discrete Fourier Transform (DFT) [34, 49]. Then, FE suppresses information of different frequencies at different time steps to decrease $\epsilon(t)$ shown in the ① term in Eq. (6). Specifically, as illustrated in Fig. 2, the information of the original image is concentrated in the low-frequency region (center area of the second column), while the noise information spans from low-frequency (center area) to high-frequency (edgearea) regions (third to fifth columns). Therefore, to remove as much frequency noise from the image as possible, the frequency suppression range gradually increases from high-frequency to low-frequency over time steps. This operator ensures that FE removes various noises present in the original image while retaining its essential information, as depicted in Fig. (3a).

Formally, denote $\boldsymbol{x} \in \mathbb{R}^{M \times N}$ as the input image and $\boldsymbol{x}^F \in \mathbb{C}^{M \times N}$ as its frequency representation, then the DFT ($\mathcal{F}$) between $\boldsymbol{x}$ and $\boldsymbol{x}^F$ is formulated as follows:

$$\boldsymbol{x}_{m,n}^F = \mathcal{F}(\boldsymbol{x}_{m,n}) = \sum_{a=0}^{M-1} \sum_{b=0}^{N-1} \boldsymbol{x}_{a,b} e^{-j2\pi(\frac{m}{M}a + \frac{n}{N}b)}, \tag{7}$$

and according to the DFT, the low-frequency parts of the image is in the center of the $\boldsymbol{x}^F$. To suppress different frequency components at different time steps, for a given time step $t$, we update $\boldsymbol{x}^{F_t}$ as follows:

$$\boldsymbol{x}^{F_t} \leftarrow \mathcal{M} \odot \boldsymbol{x}^{F_t} \tag{8}$$

where $\odot$ is element-wise multiplication. The matrix $\mathcal{M} \in \mathbb{R}^{M \times N}$ controls the scaling of different frequencies. Intuitively, $\mathcal{M}$ should be close to 0 for high-frequency components and close to 1 for low-frequency ones. In this study, we set $\mathcal{M}$ to a box window with radius $r$, defined as:

$$\mathcal{M}_{m,n} = \begin{cases} 1, & 0 \leq |m|, |n| \leq r \\ 0. & else \end{cases} \tag{9}$$

The overall function of our Frequency Encoding (FE) module at time step $t$ is then defined as:

$$\widetilde{\boldsymbol{x}}_{r_i}^t \leftarrow \mathcal{F}^{-1}(\mathcal{M}_{r_i}^t \odot \mathcal{F}^t(\boldsymbol{x})), \quad i, t \in \{1, 2, \dots, T\}, \tag{10}$$

and set

$$r_i > r_j, \quad if \quad i < j. \tag{11}$$

In summary, the proposed FE method, as described in Eq. (10), allows us to control the frequency mask radius $r$ at each time step, enabling the suppression of different frequency ranges. This effectively removes noise at various frequencies, thereby enhancing the robustness of SNNs.

## 4.3 Evolutionary leak factor to simulate the non-fixed membrane potential leak in biological nervous system

Recalling the ① term in Eq. (6), in addition to the proposed frequency encoding, the selection of the leak factor $\boldsymbol{\lambda}$ is also crucial for improving the robustness of SNNs. However, most existing work overlooks this aspect [8, 17]. They often assume that all neurons in the SNN adopt the same fixed leak factor at all time steps, which contradicts the membrane potential leak mechanism of the biological nervous system. In the biological nervous system, neuron membrane potential exhibits varying degrees of leak due to different environmental conditions and axon fibers [38, 13], aiming to enhance the processing of useful information [33], as illustrated in Fig. (1b). Therefore, in this study, we draw inspiration from the biological membrane potential leak and propose a method for training SNNs with an evolutionary membrane potential leak factor (EL).

According to ① term in Eq. (6), a smaller leak $\boldsymbol{\lambda}$ can better constrain robustness. However, excessive leak can lead to a significant loss of effective information and a decrease in original accuracy. Thus, in our approach, we aim for EL to learn the correlation between frequencies at different time steps, building upon the foundation of FE to ensure effective information utilization. We propose a trainable leak factor training scheme instead of the leak factor regularization term in our work. Specifically, leveraging the frequency-encoded input, we assign trainable leak factors to different neurons within a layer across time steps to mitigate the propagation of noise information, as shown in Fig. (3b).

Formally, the neurons in the convolutional and fully-connected layers are defined by the LIF, as illustrated in Eq. (1)(2)(3), and finally the leak factor update is computed as:

$$\boldsymbol{\lambda}_l^t = \boldsymbol{\lambda}_l^t - \eta \triangle \boldsymbol{\lambda}_l^t, \tag{12}$$

$$\triangle \boldsymbol{\lambda}_l^t = \frac{\partial \mathcal{L}}{\partial \boldsymbol{\lambda}_l^t} = \frac{\partial \mathcal{L}}{\partial \boldsymbol{O}_l^t} \cdot \frac{\partial \boldsymbol{O}_l^t}{\partial \boldsymbol{u}_l^t} \cdot \frac{\partial \boldsymbol{u}_l^t}{\partial \boldsymbol{\lambda}_l^t} = \frac{\partial \mathcal{L}}{\partial \boldsymbol{O}_l^t} \cdot \frac{\partial \boldsymbol{O}_l^t}{\partial \boldsymbol{u}_l^t} \cdot \boldsymbol{u}_l^{t-1}, \tag{13}$$

$$\mathcal{L} = \mathcal{L}_{CE}(\boldsymbol{x}, y, \boldsymbol{W}, \boldsymbol{\lambda}), \tag{14}$$

where $\frac{\partial \boldsymbol{O}_l^t}{\partial \boldsymbol{u}_l^t}$ is estimated by the surrogate gradient, $\frac{\partial \boldsymbol{O}_l^t}{\partial \boldsymbol{u}_l^t} = \frac{1}{\gamma^2} \max(0, \gamma - |\boldsymbol{u}_l^t - V_{th}|)$. $\gamma$ denotes the constraint factor that determines the sample range to activate the gradient. $\mathcal{L}_{CE}$ is the commonly used Cross-Entropy loss.

To sum up, our FEEL-SNN focuses on the ① term of Eq. (6). Here, FE serves to attenuate the impact of input noise $\epsilon(t)$ during each time step, while EL facilitates the continual learning of information correlations across varying time steps. This concerted effort enables a more effective utilization of useful information, thereby enhancing the robustness of the model.

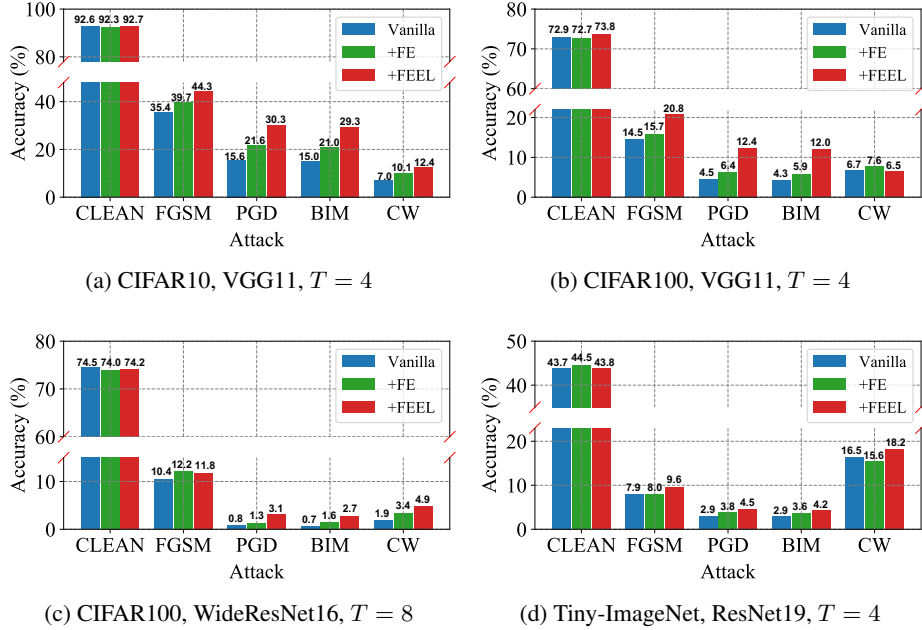

Figure 4: Performance of the proposed FE and FEEL under different **white-box attacks**. The attack perturbation $\epsilon = 4/255$ for all attacks, iterative step $k = 4$, and step size $\alpha = 0.01$ for PGD, BIM.

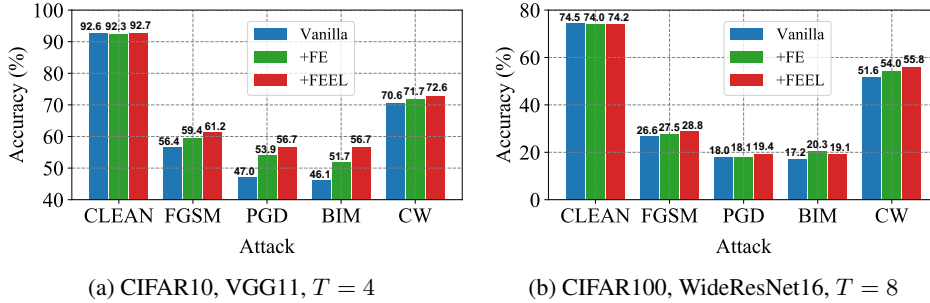

Figure 5: Performance of the proposed FE and FEEL under different **black-box attacks**. The attack perturbation $\epsilon = 4/255$ for all attacks, and iterative step $k = 4$ for PGD, BIM.

## 5 Experiments

### 5.1 Experimental settings

The datasets we used are CIFAR-10, CIFAR-100 [16], and Tiny-ImageNet [5]. The network architectures include VGG11, WideResNet16, and ResNet19. We set $\gamma = 1.0$ in surrogate gradient and threshold $V_{th} = 1.0$ in Eq. (3) following the general settings [6]. We adopted four training strategies to determine the effectiveness of the proposed FEEL method. The first is a vanilla training scheme (BPTT), directly using raw images for training [41]. The second is an adversarial training strategy, which uses examples from white-box (WB) PGD attacks ($\epsilon = 2/255$, iterative step $k = 2$) for training [12] (abbreviated as AT). The third is to add a Lipschitz penalty proposed in [8] to the weights under the adversarial training setting (abbreviated as RAT). The fourth introduces the stochastic gating mechanisms to spike firing [10] (abbreviated as StoG). For all four strategies, we test their robustness with and without the proposed FE and FEEL methods. The attack methods include adversarial attacks (i.e., FGSM [12], PGD with random start [22], BIM [18], and CW [2], for both white-box and black-box attacks) and common noise attack (i.e., gaussian noise, GN). Since the DFT ($\mathcal{F}$), IDFT ($\mathcal{F}^{-1}$) (Eq. (7) and Eq. (10)) and frequency mask operation ($\mathcal{M} \odot \boldsymbol{x}^{F_t}$, Eq. (8)) are differentiable, the FE module can be directly utilized to generate adversarial perturbations.

Table 1: Performance of the proposed FE and FEEL with different training strategies. The perturbation $\epsilon = 8/255$ for all attacks, and iterative step $k = 7$, step size $\alpha = 0.01$ for PGD, BIM. The dataset is CIFAR100 with $T = 8$, the network is VGG11. The improvement brought by our method is shown in parentheses.

| Methods | clean | GN | FGSM | PGD | BIM | CW |
|---|---|---|---|---|---|---|
| Vanilla | 72.93 | 68.93 | 4.91 | 0.16 | 0.14 | 6.53 |
| Vanilla+FE (Ours) | 72.67 (-0.26) | **69.40 (+0.47)** | 5.18 (+0.27) | 0.31 (+0.15) | 0.24 (+0.10) | **7.63 (+1.10)** |
| Vanilla+FEEL (Ours) | **73.79 (+0.86)** | 68.05(-0.88) | **9.60 (+4.69)** | **2.04 (+1.88)** | **1.81 (+1.57)** | 6.66 (+0.13) |
| AT [12] | 69.14 | 68.27 | 17.21 | 8.63 | 8.13 | 16.54 |
| AT+FE (Ours) | 69.34 (+0.20) | 68.67 (+0.40) | 17.65 (+0.44) | 8.92 (+0.29) | 8.33 (+0.20) | 21.49 (+4.95) |
| AT+FEEL (Ours) | **69.79 (+0.65)** | **69.02 (+0.75)** | **18.67 (+1.46)** | **11.07 (+2.44)** | **10.56 (+2.43)** | **21.78 (+5.24)** |
| RAT [8] | **70.03** | **69.26** | 18.88 | 8.87 | 7.93 | 20.79 |
| RAT+FE (Ours) | 69.74 (-0.29) | 68.35 (-0.91) | 18.74 (-0.14) | 9.70 (+0.83) | 8.91 (+0.98) | **27.16 (+6.37)** |
| RAT+FEEL (Ours) | 69.80 (-0.23) | 68.46 (-0.80) | **19.08 (+0.20)** | **12.36 (+3.49)** | **11.96 (+4.03)** | 25.52 (+4.73) |
| StoG [10] | 72.22 | 61.63 | 5.92 | 0.26 | 0.20 | 19.87 |
| StoG+FE (Ours) | **73.13 (+0.91)** | **67.65 (+6.02)** | 6.95 (+1.03) | 0.22 (-0.04) | 0.25 (+0.05) | 23.02 (+3.15) |
| StoG+FEEL (Ours) | 72.13 (-0.09) | 65.96 (+4.33) | **9.15 (+3.23)** | **0.55 (+0.29)** | **0.31 (+0.11)** | **24.79 (+4.92)** |
| AT+StoG | 69.24 | 63.35 | 19.64 | 9.77 | 3.23 | 44.79 |
| AT+StoG+FE (Ours) | 69.45 (+0.21) | **68.83 (+5.48)** | **20.06 (+0.42)** | 10.69 (+0.92) | 3.24 (+0.01) | 38.56 (-6.23) |
| AT+StoG+FEEL (Ours) | **69.53 (+0.29)** | 68.47 (+5.12) | 18.27 (-1.37) | **11.52 (+1.75)** | **3.90 (+0.67)** | **45.18 (+0.39)** |
| RAT+StoG | 69.12 | 68.37 | 29.25 | 15.43 | 6.91 | 32.08 |
| RAT+StoG+FE (Ours) | 68.97 (-0.15) | **68.52 (+0.15)** | 31.65 (+2.40) | 17.49 (+2.06) | 8.57 (+1.66) | 47.16 (+15.08) |
| RAT+StoG+FEEL (Ours) | **69.97 (+0.85)** | 68.15 (-0.22) | **31.68 (+2.43)** | **18.07 (+2.64)** | **8.89 (+1.98)** | **50.56 (+18.48)** |

Therefore, the adversarial perturbations are applied to the image domain before FE. In our study, for CIFAR10 and CIFAR100 dataset with $T = 4$, we set $r = [16, 14, 12, 10]$. For CIFAR10 and CIFAR100 dataset with $T = 8$, we set $r = [16, 14, 12, 10, 8, 6, 4, 2]$. For Tiny-ImageNet with $T = 4$, we set $r = [32, 30, 28, 26]$. The impact of the frequency masking radius $r$ on robustness is detailed in Section 5.3. More detailed experimental settings can be found in Appendix A.3.

## 5.2 Overall performance for various attack types

**White-box attack.** First, we integrate the proposed FE and FEEL methods into the standard training (vanilla) of SNNs. We present experimental results for our method on various datasets (*i.e.*, CIFAR-10, CIFAR-100, and Tiny-ImageNet) using different networks (*i.e.*, VGG11, WideResNet16, and ResNet19) under white-box attacks, as summarized in Fig. 4. Our findings demonstrate that across all attacks, FE and FEEL can enhance model robustness and maintain the original accuracy. Specifically, on VGG11 with CIFAR-10, compared to the vanilla method (shown in blue bar), FEEL enhances model robustness by up to 15% and 6% against PGD and CW attacks, respectively, at time step 4. Similar trends are observed across other datasets and networks. Moreover, it is clear from the Fig. 4 that simple FE application can effectively improve the robustness of SNN, and EL can further effectively improve the robustness based on FE.

**Black-box attack.** We utilize a model trained with a different seed to generate perturbed images for black-box attacks. The efficacy of our FE and FEEL method under various attacks is illustrated in Fig. 5. Across all models and datasets, the same observation can be obtained as the white-box performance. FE and FEEL consistently outperform vanilla training. Notably, with $T = 8$, FEEL enhances robustness (attacked by CW) by up to 4.27% compared to the vanilla approach, when on WideResNet16 with CIFAR100.

**Comparison with state-of-the-art work on robustness of SNN.** To further evaluate the effectiveness of our FE and FEEL methods, we compare it with state-of-the-art (SOTA) robust SNN methods, namely AT [12], RAT [8], and StoG [10] in Tab. 1. From Tab. 1, we observe that FE and FEEL can enhance the original accuracy and robustness of these SOTA methods. For example, when under attack by PGD, SNN-RAT improves the robustness of the original model (Vanilla) from 0.16% to 8.87%, our FE (RAT+FE) enhances the robustness of RAT (RAT) to 9.70%. FEEL further enhances the robustness of RAT (RAT+FEEL) to 12.36%. These experimental results underscore the effectiveness of our FE and FEEL methods.

More experimental results of our FE, EL, and FEEL with different time steps, datasets, and networks can be found in Appendix. A.4.

## 5.3 Ablation study

**Performance under different $\epsilon$ and iterative step $k$.** We plot the accuracy of the white-box and black-box scenarios under PGD attack with varying $\epsilon$ and iterative step $k$ in Fig. 6 and Fig. 7, respectively. The results indicate that the accuracy of our FEEL models decreases slowly compared to that of vanilla models.

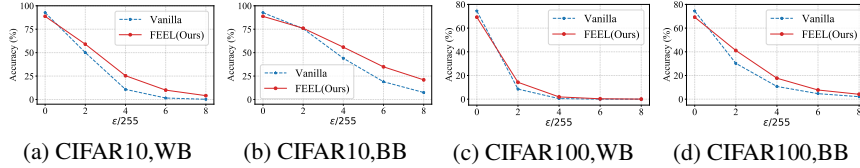

(a) CIFAR10,WB     (b) CIFAR10,BB     (c) CIFAR100,WB     (d) CIFAR100,BB

Figure 6: Performance of the white-box (WB) and black-box (BB) scenarios under PGD attack with different perturbation $\epsilon$, the iterative step $k = 4$, the network is VGG11.

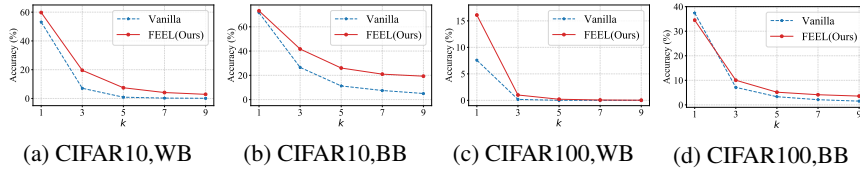

(a) CIFAR10,WB     (b) CIFAR10,BB     (c) CIFAR100,WB     (d) CIFAR100,BB

Figure 7: Performance of the white-box (WB) and black-box (BB) scenarios under PGD attack with different iterative step $k$, the perturbation $\epsilon = 4/255$, the network is VGG11.

**Rationality of FE method.** To further verify the effectiveness of FE which crops information from high-frequency to low-frequency over time steps, we compare it with an alternative strategy, Inverse-FE (IFE), which crops information from low-frequency to high-frequency over time steps. As shown in Tab. 2, IFE causes a significant drop in clean accuracy (64.81% vs. vanilla 92.64%). This demonstrates that a substantial amount of valid information is lost, verifying that valid information is concentrated in the low-frequency area. In contrast, FE not only effectively removes noise (21.56% vs. vanilla 15.59% when under PGD attack) but also minimizes the loss of valid information (92.26% vs. vanilla 92.64%).

Table 2: Performance (%) of the proposed Frequency Encoding (FE) and the alternative strategy Inverse-FE (IFE). The perturbation $\epsilon = 4/255$ for all attacks, and iterative step $k = 4$, step size $\alpha = 0.01$ for PGD. The dataset is CIFAR10 with time step $T = 4$, the network is VGG11.

| Method | Clean | GN | FGSM | PGD | BIM | CW |
|---|---|---|---|---|---|---|
| Vanilla | **92.64** | 91.28 | 35.47 | 15.59 | 14.95 | 6.92 |
| IFE | 64.81 | 64.48 | 12.33 | 4.44 | 4.25 | 4.18 |
| FE | 92.26 | **92.02** | **39.67** | **21.56** | **21.05** | **10.12** |

**Effect of frequency masking radius $r$ on robustness.** We investigate the frequency masking radius $r$ on robustness to SNNs, defined in Eq. (9), which governs the degree of frequency suppression at each time step. We present three different strategies to illustrate the superiority of our method as shown in Tab. 3. The first strategy employs a direct encoding method, wherein the frequency information outside a fixed radius $r$ is removed from each original image, followed by T-step image replication (using a different $r$ for each image, akin to data augmentation), as depicted in the first row of Tab. 3. The second strategy utilizes FE but with a uniform radius $r$ across all time steps, as shown in rows 2 to 5 of Tab. 3. The third strategy is the FE method proposed in this study, where a different $r$ is applied to each time step to remove as many noise frequencies as possible, as depicted in the last row of Tab. 3. From Tab. 3, we observe that across various $r$ selections, our method outperforms the first and second strategies in all attack scenarios.

**Rationality of EL method.** According to Eq. (2), the leak factor controls the residual membrane potential between time steps. A smaller leak factor may lead to a weakened temporal modeling capability of the SNN, leading to a decline in network performance [28]. Considering the leak factor's

Table 3: Effect of frequency masking radius $r$ on robustness. The attack is PGD with perturbation $\epsilon = 4/255$, iterative step $\alpha = 0.01$, and iterative step $k = 4$. The dataset is CIFAR10 with $T = 4$, the network is VGG11. $r_0 = 16, r_1 = 14, r_2 = 12, r_3 = 10, r_4 = 6$

| Encode | r | clean | GN | FGSM | PGD | BIM | CW |
|---|---|---|---|---|---|---|---|
| Direct | $[r_0]_4, [r_1]_4, [r_2]_4, [r_3]_4$ | 70.88 | 69.73 | 14.43 | 4.33 | 4.19 | 6.21 |
| FE | $[r_4]_4$ | 62.01 | 60.73 | 9.47 | 2.28 | 2.17 | 4.79 |
| FE | $[r_3]]_4$ | 68.78 | 67.55 | 13.62 | 4.74 | 4.35 | 6.37 |
| FE | $[r_2]_4$ | 69.96 | 69.26 | 14.60 | 5.38 | 5.20 | 6.87 |
| FE | $[r_1]_4$ | 70.95 | 70.39 | 15.72 | 5.41 | 5.22 | 7.45 |
| FE (Ours) | $[r_0, r_1, r_2, r_3]$ | **71.40** | **70.59** | **16.80** | **6.89** | **6.62** | **8.09** |

dual role in original information transmission (Eq. (2)) and robustness enhancement (Eq. (6)), we propose EL. The EL dynamically learns the optimal robustness leak factor across different time steps and neurons, which also increases the expression capability of SNN, helping maintain clean accuracy and improving robustness. We further compare EL with two alternative strategies. The first strategy sets all leak factors to 0. The second strategy, termed Reg-EL (REL), adds L2 regularization to the EL to further constrain the leak factor. As shown in Tab. 6 in Appendix. A.4, a small leak factor significantly reduces clean accuracy (vanilla 92.64% vs. REL 88.52% vs. EL with $\lambda = 0.0$ at 81.76%), consistent with analysis above. Besides, a small leak factor does increase the robustness of SNN (*e.g.*, under PGD attack, EL with $\lambda = 0.0$ is 63.80%, REL is 29.98%, compared to vanilla 15.59%). This also aligns with the proposed robustness framework (Eq. (6)) by demonstrating that controlling the leak factor improves robustness. And our EL method ensures improvements in both robustness and original accuracy (*e.g.*, the PGD defense accuracy of EL is 30.27%, compared to 15.59% for vanilla, and the clean accuracy of EL is 92.73%, compared to 92.64% for vanilla). In Appendix. A.4, we further verify the proposed EL does not destroy the impact of other terms in Eq. (6) on the robustness of SNN.

All the above experimental results illustrate the rationality and effectiveness of our method.

## 6 Conclusions and Discussions

**Conclusion:** In this study, drawing inspiration from the selective visual attention and dynamic membrane potential leak observed in biological nervous systems, we introduce a robust SNN with Frequency Encoding and Evolutionary membrane potential Leak factor (FEEL-SNN). Specifically, our approach theoretically presents a unified framework for SNN robustness, demonstrating that refining the encoding technique and evolving the membrane potential leak factor can enhance SNN robustness. Then we propose a novel image encoding method for SNNs, termed frequency encoding (FE). FE captures information of varying frequencies at different time steps, which preserves the original information while suppressing different frequency range noises, effectively filtering out image noise. Building upon FE, we propose an evolutionary leak factor (EL). EL ensures that different neurons in the network learn the optimal robustness leak factor at different time steps. It facilitates the continual learning of information correlations across varying time steps, enabling more effective utilization of pertinent information and thereby enhancing SNN robustness. Experimental results validate that both our FE and EL methods can effectively improve the robustness of SNN to different noises, and can be used in conjunction with other methods to improve the robustness further.

**Limitation:** Our focus has been primarily on static datasets. In future work, how to propose a reliable and effective encoding method for the DVS datasets is a topic worthy of study.

**Broader Impact:** In neuroscience, the selective visual attention and non-fixed membrane potential leak are considered to contribute to the robustness of biological nervous systems. By using SNN as a research tool, computational modeling of biological nervous systems can be further facilitated. We can contribute valuable insights to ongoing discussions in neuroscience regarding robustness.

## 7 Acknowledgement

This work was supported in part by the STI 2030 Major Projects under Grant 2021ZD0200400, in part by the National Natural Science Foundation of China (61925603, 62376247, U20A20220, and 62334014), and in part by the grants from Key R&D Program of Zhejiang (2022C01048).

# References

[1] Rémi Bernhard, Pierre-Alain Moëllic, Martial Mermillod, Yannick Bourrier, Romain Cohendet, Miguel Solinas, and Marina Reyboz. Impact of spatial frequency based constraints on adversarial robustness. In *2021 International Joint Conference on Neural Networks (IJCNN)*, pages 1–8. IEEE, 2021.

[2] Nicholas Carlini and David Wagner. Towards evaluating the robustness of neural networks. In *2017 ieee symposium on security and privacy (sp)*, pages 39–57. Ieee, 2017.

[3] Sayeed Shafayet Chowdhury, Chankyu Lee, and Kaushik Roy. Towards understanding the effect of leak in spiking neural networks. *Neurocomputing*, 464:83–94, 2021.

[4] Michael V DeBole, Brian Taba, Arnon Amir, Filipp Akopyan, Alexander Andreopoulos, William P Risk, Jeff Kusnitz, Carlos Ortega Otero, Tapan K Nayak, Rathinakumar Appuswamy, et al. Truenorth: Accelerating from zero to 64 million neurons in 10 years. *Computer*, 52(5):20–29, 2019.

[5] Jia Deng, Wei Dong, Richard Socher, Li-Jia Li, Kai Li, and Li Fei-Fei. Imagenet: A large-scale hierarchical image database. In *2009 IEEE conference on computer vision and pattern recognition*, pages 248–255. Ieee, 2009.

[6] Shikuang Deng, Yuhang Li, Shanghang Zhang, and Shi Gu. Temporal efficient training of spiking neural network via gradient re-weighting. In *International Conference on Learning Representations*, 2021.

[7] Robert Desimone and John Duncan. Neural mechanisms of selective visual attention. *Annual review of neuroscience*, 18(1):193–222, 1995.

[8] Jianhao Ding, Tong Bu, Zhaofei Yu, Tiejun Huang, and Jian Liu. Snn-rat: Robustness-enhanced spiking neural network through regularized adversarial training. *Advances in Neural Information Processing Systems*, 35:24780–24793, 2022.

[9] Jianhao Ding, Zhaofei Yu, Tiejun Huang, and Jian K Liu. Spike timing reshapes robustness against attacks in spiking neural networks. *arXiv preprint arXiv:2306.05654*, 2023.

[10] Jianhao Ding, Zhaofei Yu, Tiejun Huang, and Jian K Liu. Enhancing the robustness of spiking neural networks with stochastic gating mechanisms. In *Proceedings of the AAAI Conference on Artificial Intelligence*, volume 38, pages 492–502, 2024.

[11] Rida El-Allami, Alberto Marchisio, Muhammad Shafique, and Ihsen Alouani. Securing deep spiking neural networks against adversarial attacks through inherent structural parameters. In *2021 Design, Automation & Test in Europe Conference & Exhibition (DATE)*, pages 774–779. IEEE, 2021.

[12] Ian J Goodfellow, Jonathon Shlens, and Christian Szegedy. Explaining and harnessing adversarial examples. *arXiv preprint arXiv:1412.6572*, 2014.

[13] Nathan W Gouwens, Staci A Sorensen, Jim Berg, Changkyu Lee, Tim Jarsky, Jonathan Ting, Susan M Sunkin, David Feng, Costas A Anastassiou, Eliza Barkan, et al. Classification of electrophysiological and morphological neuron types in the mouse visual cortex. *Nature neuroscience*, 22(7):1182–1195, 2019.

[14] Jianing Han, Ziming Wang, Jiangrong Shen, and Huajin Tang. Symmetric-threshold relu for fast and nearly lossless ann-snn conversion. *Machine Intelligence Research*, 20(3):435–446, 2023.

[15] Yangfan Hu, Qian Zheng, Xudong Jiang, and Gang Pan. Fast-snn: Fast spiking neural network by converting quantized ann. *IEEE Transactions on Pattern Analysis and Machine Intelligence*, 2023.

[16] Alex Krizhevsky, Geoffrey Hinton, et al. Learning multiple layers of features from tiny images. 2009.

[17] Souvik Kundu, Massoud Pedram, and Peter A Beerel. Hire-snn: Harnessing the inherent robustness of energy-efficient deep spiking neural networks by training with crafted input noise. In *Proc. of International Conference on Computer Vision*, pages 5209–5218, 2021.

[18] Alexey Kurakin, Ian J Goodfellow, and Samy Bengio. Adversarial examples in the physical world. *arXiv preprint arXiv:1607.02533*, 2016.

[19] Zhanfeng Liao, Yan Liu, Qian Zheng, and Gang Pan. Spiking nerf: Representing the real-world geometry by a discontinuous representation. In *Proceedings of the AAAI Conference on Artificial Intelligence*, volume 38, pages 13790–13798, 2024.

[20] De Ma, Xiaofei Jin, Shichun Sun, Yitao Li, Xundong Wu, Youneng Hu, Fangchao Yang, Huajin Tang, Xiaolei Zhu, Peng Lin, et al. Darwin3: a large-scale neuromorphic chip with a novel isa and on-chip learning. *National Science Review*, 11(5):nwae102, 2024.

[21] Wolfgang Maass. Networks of spiking neurons: the third generation of neural network models. *Neural networks*, 10(9):1659–1671, 1997.

[22] Aleksander Madry, Aleksandar Makelov, Ludwig Schmidt, Dimitris Tsipras, and Adrian Vladu. Towards deep learning models resistant to adversarial attacks. In *International Conference on Learning Representations*, 2018.

[23] Shishira R Maiya, Max Ehrlich, Vatsal Agarwal, Ser-Nam Lim, Tom Goldstein, and Abhinav Shrivastava. A frequency perspective of adversarial robustness. *arXiv preprint arXiv:2111.00861*, 2021.

[24] Alberto Marchisio, Giorgio Nanfa, Faiq Khalid, Muhammad Abdullah Hanif, Maurizio Martina, and Muhammad Shafique. Is spiking secure? a comparative study on the security vulnerabilities of spiking and deep neural networks. In *2020 International Joint Conference on Neural Networks (IJCNN)*, pages 1–8. IEEE, 2020.

[25] Emre O Neftci, Hesham Mostafa, and Friedemann Zenke. Surrogate gradient learning in spiking neural networks: Bringing the power of gradient-based optimization to spiking neural networks. *IEEE Signal Processing Magazine*, 36(6):51–63, 2019.

[26] Jing Pei, Lei Deng, Sen Song, Mingguo Zhao, Youhui Zhang, Shuang Wu, Guanrui Wang, Zhe Zou, Zhenzhi Wu, Wei He, et al. Towards artificial general intelligence with hybrid tianjic chip architecture. *Nature*, 572(7767):106–111, 2019.

[27] Chongli Qin, James Martens, Sven Gowal, Dilip Krishnan, Krishnamurthy Dvijotham, Alhussein Fawzi, Soham De, Robert Stanforth, and Pushmeet Kohli. Adversarial robustness through local linearization. *Advances in neural information processing systems*, 32, 2019.

[28] Nitin Rathi and Kaushik Roy. Diet-snn: A low-latency spiking neural network with direct input encoding and leakage and threshold optimization. *IEEE Transactions on Neural Networks and Learning Systems*, 34(6):3174–3182, 2021.

[29] Kevin Roth, Yannic Kilcher, and Thomas Hofmann. Adversarial training is a form of data-dependent operator norm regularization. *Advances in Neural Information Processing Systems*, 33:14973–14985, 2020.

[30] Kaushik Roy, Akhilesh Jaiswal, and Priyadarshini Panda. Towards spike-based machine intelligence with neuromorphic computing. *Nature*, 575(7784):607–617, 2019.

[31] Saima Sharmin, Priyadarshini Panda, Syed Shakib Sarwar, Chankyu Lee, Wachirawit Ponghiran, and Kaushik Roy. A comprehensive analysis on adversarial robustness of spiking neural networks. In *2019 International Joint Conference on Neural Networks (IJCNN)*, pages 1–8. IEEE, 2019.

[32] Saima Sharmin, Nitin Rathi, Priyadarshini Panda, and Kaushik Roy. Inherent adversarial robustness of deep spiking neural networks: Effects of discrete input encoding and non-linear activations. In *European Conference on Computer Vision*, pages 399–414. Springer, 2020.

[33] Yousheng Shu, Andrea Hasenstaub, Alvaro Duque, Yuguo Yu, and David A McCormick. Modulation of intracortical synaptic potentials by presynaptic somatic membrane potential. *Nature*, 441(7094):761–765, 2006.

[34] Duraisamy Sundararajan. *The discrete Fourier transform: theory, algorithms and applications*. World Scientific, 2001.

[35] Christian Szegedy, Wojciech Zaremba, Ilya Sutskever, Joan Bruna, Dumitru Erhan, Ian Goodfellow, and Rob Fergus. Intriguing properties of neural networks. In *International Conference on Learning Representations*, 2014.

[36] Doron Tal and Eric L Schwartz. Computing with the leaky integrate-and-fire neuron: logarithmic computation and multiplication. *Neural computation*, 9(2):305–318, 1997.

[37] Amirhossein Tavanaei, Masoud Ghodrati, Saeed Reza Kheradpisheh, Timothée Masquelier, and Anthony Maida. Deep learning in spiking neural networks. *Neural networks*, 111:47–63, 2019.

[38] Corinne Teeter, Ramakrishnan Iyer, Vilas Menon, Nathan Gouwens, David Feng, Jim Berg, Aaron Szafer, Nicholas Cain, Hongkui Zeng, Michael Hawrylycz, et al. Generalized leaky integrate-and-fire models classify multiple neuron types. *Nature communications*, 9(1):709, 2018.

[39] Cheng Wang, Chankyu Lee, and Kaushik Roy. Noise resilient leaky integrate-and-fire neurons based on multi-domain spintronic devices. *Scientific Reports*, 12(1):8361, 2022.

[40] Ziming Wang, Runhao Jiang, Shuang Lian, Rui Yan, and Huajin Tang. Adaptive smoothing gradient learning for spiking neural networks. In *International Conference on Machine Learning*, pages 35798–35816. PMLR, 2023.

[41] Yujie Wu, Lei Deng, Guoqi Li, and Luping Shi. Spatio-temporal backpropagation for training high-performance spiking neural networks. *Frontiers in neuroscience*, 12:323875, 2018.

[42] Hayat Yedjour and Dounia Yedjour. A spatiotemporal energy model based on spiking neurons for human motion perception. *Cognitive Neurodynamics*, pages 1–15, 2024.

[43] Dong Yin, Raphael Gontijo Lopes, Jon Shlens, Ekin Dogus Cubuk, and Justin Gilmer. A fourier perspective on model robustness in computer vision. *Advances in Neural Information Processing Systems*, 32, 2019.

[44] Takashi Yoshida and Kenichi Ohki. Natural images are reliably represented by sparse and variable populations of neurons in visual cortex. *Nature communications*, 11(1):872, 2020.

[45] Alberto Zani and Alice Mado Proverbio. Selective attention to spatial frequency gratings affects visual processing as early as 60 msec. poststimulus. *Perceptual and motor skills*, 109(1):140–158, 2009.

[46] Ming ZHANG, Zonghua Gu, and Gang Pan. A survey of neuromorphic computing based on spiking neural networks. *Chinese Journal of Electronics*, 27(4):667–674, 2018.

[47] Wenrui Zhang and Peng Li. Temporal spike sequence learning via backpropagation for deep spiking neural networks. *Advances in Neural Information Processing Systems*, 33:12022–12033, 2020.

[48] Weixing Zhang, Zongrui Li, De Ma, Huajin Tang, Xudong Jiang, Qian Zheng, and Gang Pan. Spiking gs: Towards high-accuracy and low-cost surface reconstruction via spiking neuron-based gaussian splatting. *arXiv preprint arXiv:2410.07266*, 2024.

[49] Zhendong Zhang, Cheolkon Jung, and Xiaolong Liang. Adversarial defense by suppressing high-frequency components. *arXiv preprint arXiv:1908.06566*, 2019.

# A Appendix

## A.1 The attacks we used in this study

Given a classification model $f$ with dataset $(\boldsymbol{x}, y_{true})$, where $\boldsymbol{x}$ is the clean image and $y_{true}$ is the corresponding correct label. The formulations of the attacks we used in this study are described as follows:

**FGSM.** FGSM aims to perturb the original data $\boldsymbol{x}$ along the sign direction of the gradient on loss function with one step to increase the perturbed linear output, thus fool the network, it can be formalized as follows:

$$\hat{\boldsymbol{x}} = \boldsymbol{x} + \epsilon\,\mathrm{sign}(\nabla_{\boldsymbol{x}}\mathcal{L}(f(\boldsymbol{x}), y_{true})), \tag{15}$$

where $\mathrm{sign}(\cdot)$ is an odd mathematical function that extracts the sign of a real number.

**PGD.** PGD attack is the iterative variant of FGSM. It first starts from a random perturbation in the $L_p$-norm constraint around the original sample $\boldsymbol{x}$, then takes a gradient iteration step in the sign direction to achieve the greatest loss output, it can be formalized as follows:

$$\hat{\boldsymbol{x}}^0 = \boldsymbol{x} + \mathrm{U}(-\epsilon, +\epsilon), \tag{16}$$

$$\hat{\boldsymbol{x}}^{k+1} = \mathrm{Clip}_{\boldsymbol{x},\epsilon}\{\hat{\boldsymbol{x}}^k + \alpha \cdot \mathrm{sign}(\nabla_{\hat{\boldsymbol{x}}^k}\mathcal{L}(f(\hat{\boldsymbol{x}}^k), y_{true}))\}, \tag{17}$$

where $k$ is the iterative step, $\alpha$ is step size for each attack iteration, $\epsilon$ controls the perturbation level. $\mathrm{U}(\cdot)$ is a uniform function, $\mathrm{Clip}_{x,\epsilon}\{\boldsymbol{x}\}$ is the function which performs per-pixel clipping of the image $\hat{\boldsymbol{x}}$, so the result will be in $L_\infty$-norm $\epsilon$-neighbourhood of the original image $\boldsymbol{x}$.

**BIM.** Both BIM and PGD attacks are iterative attacks. Different from PGD attacks, BIM updates the adversarial samples starting from the original image.

**CW.** CW attack is different from previous gradient-based attack methods. It is based on model optimization to generate adversarial samples. Its optimization function is as follows:

$$minimize||\frac{1}{2}(\tanh(\boldsymbol{W}) + 1) - \boldsymbol{x}||_2^2 + c \cdot f(\tanh(\boldsymbol{W}) + 1), \tag{18}$$

where $f$ defined as

$$f(\hat{\boldsymbol{x}}) = \max(\max\{Z(\hat{\boldsymbol{x}})_i : i \neq j\} - Z(\hat{\boldsymbol{x}})_j, -k), \tag{19}$$

where $c$ is a parameter to control the perturbation, $Z(\cdot)_i$ represents the logits output on label $y_i$.

## A.2 Proof for Theorem 1

Proof for Theorem 1 is given as follows:

*Proof.* By applying the spatial-temporal backpropagation (STBP) rule [41], we have

$$\frac{\partial\mathcal{L}}{\partial\boldsymbol{x}^t} = \frac{1}{L}\sum_{l=1}^{L}[(\frac{\partial\mathcal{L}}{\partial\boldsymbol{O}_l^T}\frac{\partial\boldsymbol{O}_l^T}{\partial\boldsymbol{u}_l^T})(\frac{\partial\boldsymbol{u}_l^T}{\partial\boldsymbol{u}_l^{T-1}}\cdots\frac{\partial\boldsymbol{u}_l^{t+1}}{\partial\boldsymbol{u}_l^t})(\frac{\partial\boldsymbol{u}_l^t}{\partial\boldsymbol{O}_{l-1}^t}\frac{\partial\boldsymbol{O}_{l-1}^t}{\partial\boldsymbol{u}_{l-1}^t}\cdots\frac{\partial\boldsymbol{u}_2^t}{\partial\boldsymbol{O}_1^t}\frac{\partial\boldsymbol{O}_1^t}{\partial\boldsymbol{u}_1^t})\cdot\frac{\partial\boldsymbol{u}_1^t}{\partial\boldsymbol{x}^t}],$$

$$= \frac{1}{L}\sum_{l=1}^{L}[\prod_{k=t}^{T}\boldsymbol{\lambda}_l^k \cdot \prod_{q=2}^{l}\boldsymbol{W}_{q-1,q}\cdot(\frac{\partial\mathcal{L}}{\partial\boldsymbol{O}_l^T}\frac{\partial\boldsymbol{O}_l^T}{\partial\boldsymbol{u}_l^T}\frac{\partial\boldsymbol{O}_{l-1}l^t}{\partial\boldsymbol{u}_{l-1}^t}\cdots\frac{\partial\boldsymbol{O}_1^t}{\partial\boldsymbol{u}_1^t})] \tag{20}$$

$$= \frac{1}{L}\sum_{l=1}^{L}[\prod_{k=t}^{T}\boldsymbol{\lambda}_l^k \cdot \prod_{q=2}^{l}\boldsymbol{W}_{q-1,q}\cdot\prod_{v=1}^{l}\frac{\partial\boldsymbol{O}_v^t}{\partial\boldsymbol{u}_v^t}\cdot\frac{\partial\mathcal{L}}{\partial\boldsymbol{O}_l^T}]$$

Therefore,

$$\min\sum_t|\epsilon(t)\odot\frac{\partial\mathcal{L}}{\partial\boldsymbol{x}^t}|_1 = \min\sum_t|\frac{1}{L}\sum_{l=1}^{L}[(\underbrace{\prod_{k=t}^{T}\epsilon(t)\odot\boldsymbol{\lambda}_l^k}_{①})\cdot\underbrace{\prod_{q=2}^{l}\boldsymbol{W}_{q-1,q}}_{②}\cdot\underbrace{\prod_{v=1}^{l}\frac{\partial\boldsymbol{O}_v^t}{\partial\boldsymbol{u}_v^t}}_{③}\cdot\frac{\partial\mathcal{L}}{\partial\boldsymbol{O}_l^T}]|_1, \tag{21}$$

### A.3 More experimental settings

In our work, the training process lasts for 300 epochs for all experiments. Batch normalization are used in the network to overcome the gradient vanishing or explosion. SGD optimizer is deployed, and the initial learning rate is set to 0.1. The learning rate uses a cosine annealing schedule with $T_{max}$ equaling the max number of epochs. All the experiments are conducted on the PyTorch platform on NVIDIA RTX 3090.

### A.4 More experimental results

In Tab. 4, we integrate the proposed FE and FEEL methods into the standard training (vanilla) of SNNs. The same as the results in the main paper, we can see from Tab. 4 that in different datasets (*i.e.*, CIFAR-10, CIFAR-100, and Tiny-ImageNet), different networks (*i.e.*, VGG11, WideResNet16, and ResNet19) and different time steps ($T = 4, 8$), both FE and FEEL can effectively and stably improve the robustness of the vanilla model to different attacks. This further verify the effectiveness of our method.

Table 4: Performance of the proposed FE and FEEL under different **white-box attacks**. The attack perturbation $\epsilon = 4/255$ for all attacks, iterative step $k = 4$, and step size $\alpha = 0.01$ for PGD, BIM. 'WR16' represents WideResNet16 and 'R19' represents ResNet19.

| Datasets | networks | T | method | clean | GN | FGSM | PGD | BIM | CW |
|---|---|---|---|---|---|---|---|---|---|
| CIFAR10 | VGG11 | 4 | Vanilla | 92.64 | 92.28 | 35.47 | 15.59 | 14.95 | 6.92 |
| | | | +FE (Ours) | 92.26 | 92.02 | 39.67 | 21.56 | 21.05 | 10.12 |
| | | | +FEEL (Ours) | **92.73** | **92.59** | **44.25** | **30.27** | **29.34** | **12.39** |
| | | 8 | Vanilla | **93.35** | **92.72** | 34.15 | 13.12 | 12.29 | 7.11 |
| | | | +FE (Ours) | 92.92 | 92.04 | 38.34 | 19.45 | 18.59 | 8.94 |
| | | | +FEEL (Ours) | 93.29 | 92.12 | **44.96** | **28.35** | **27.18** | **12.19** |
| | WR16 | 8 | Vanilla | **94.19** | **92.24** | 23.01 | 1.43 | 1.28 | 3.48 |
| | | | +FE (Ours) | 94.15 | 92.40 | 24.59 | 2.11 | 1.87 | 3.69 |
| | | | +FEEL (Ours) | 91.65 | 91.13 | **27.58** | **5.88** | **5.54** | **3.92** |
| CIFAR100 | VGG11 | 4 | Vanilla | 72.11 | **71.30** | 15.26 | 5.30 | 5.01 | 7.01 |
| | | | +FE (Ours) | 71.40 | 70.59 | 16.80 | 6.89 | 6.62 | **8.09** |
| | | | +FEEL (Ours) | **72.40** | 70.63 | **23.63** | **14.07** | **13.62** | 7.78 |
| | | 8 | Vanilla | 72.93 | 71.75 | 14.51 | 4.48 | 4.26 | 6.66 |
| | | | +FE (Ours) | 72.67 | 71.78 | 15.70 | 6.37 | 5.87 | **7.63** |
| | | | +FEEL (Ours) | **73.79** | **73.28** | **20.78** | **12.35** | **12.01** | 6.53 |
| | WR16 | 8 | Vanilla | **74.51** | 68.24 | 10.41 | 0.81 | 0.73 | 1.87 |
| | | | +FE (Ours) | 73.97 | **68.71** | **12.24** | 1.31 | 1.58 | 3.44 |
| | | | +FEEL (Ours) | 74.23 | 66.76 | 11.82 | **3.05** | **2.67** | **4.85** |
| Tiny-ImageNet | R19 | 4 | Vanilla | 43.72 | 42.64 | 7.98 | 2.91 | 2.94 | 16.48 |
| | | | +FE (Ours) | **44.46** | **44.22** | 8.01 | 3.79 | 3.62 | 15.63 |
| | | | +FEEL (Ours) | 43.83 | 43.34 | **9.59** | **4.53** | **4.22** | **18.21** |

**Performance when using the EL method alone.** We have included the performance of our EL combined with the vanilla and SOTA robust methods in Tab. 5. Combined with Tab. 1. It is evident that both FE and EL effectively enhance the robustness of the original methods, with FEEL further improving robustness on this foundation. For instance, under a PGD attack, the original AT method achieves 8.63% accuracy, while our FE increases robustness to 9.70%, EL to 11.15%, and FEEL to 12.36%. This illustrates the effectiveness of each module of our method.

**EL does not destroy the impact of other terms in Eq. (6) on the robustness of SNN.** We would like to discuss how the leak factor affects other terms in Eq. (6) in two cases: 1) leak factor $\lambda$ as a hyperparameter predefined before neural network training and 2) leak factor $\lambda$ as a learnable parameter during neural network training (the proposed EL implementation).

1). In the first case, $\lambda$ is a fixed number during neural network training (similar to $\epsilon$ in ① term). Hence, it will not affect other terms in Eq. (6)) at all. To validate the correctness of our theoretical framework (*i.e.*, smaller ① term results in less perturbation in the output), we conduct additional experiments, *i.e.*, training different neural networks with different fixed $\lambda$ (keep the remaining settings exactly the same as that reported in experimental settings in the main paper). As results shown in Tab. 6, a smaller $\lambda$ results in a more robust model, indicating that smaller ① term results in less perturbation in

Table 5: Performance (%) of EL under different attacks. * represents black-box attack performance. 'WR16' represents WideResNet16 and 'R19' represents ResNet19. 'GP' represents gradient penalty regularization.

| Setting | method | clean | FGSM | PGD | BIM | CW |
|---|---|---|---|---|---|---|
| $\epsilon = 4/255, k = 4, \alpha = 0.01$ | | | | | | |
| CIFAR10, VGG11, $T4$ | Vanilla+EL | 91.24 | 42.63 | 27.69 | 26.41 | 11.43 |
| CIFAR10, VGG11, $T4$ | Vanilla+EL* | 92.24 | 60.26 | 55.37 | 56.86 | 71.91 |
| CIFAR10, VGG11, $T8$ | Vanilla+EL | 92.69 | 41.22 | 25.39 | 23.97 | 9.95 |
| CIFAR10, WR16, $T8$ | Vanilla+EL | 91.57 | 23.89 | 3.11 | 2.78 | 3.06 |
| CIFAR100, VGG11, $T4$ | Vanilla+EL | 70.13 | 21.03 | 11.89 | 11.44 | 6.79 |
| CIFAR100, VGG11, $T8$ | Vanilla+EL | 72.41 | 22.03 | 12.95 | 12.16 | 6.98 |
| CIFAR100, WR16, $T8$ | Vanilla+EL | 73.20 | 10.87 | 2.14 | 2.12 | 3.86 |
| CIFAR100, WR16, $T8$ | Vanilla+EL* | 74.20 | 29.22 | 19.53 | 19.0 | 55.6 |
| Tiny-ImageNet, R19, $T4$ | Vanilla+EL | 45.15 | 9.69 | 4.43 | 4.27 | 21.95 |
| CIFAR10, VGG11, $T4$ | GP+EL | 90.53 | 41.72 | 26.33 | 25.09 | 20.21 |
| $\epsilon = 8/255, k = 7, \alpha = 0.01$ | | | | | | |
| CIFAR100, VGG11, $T8$ | Vanilla+EL | 71.41 | 9.16 | 1.29 | 1.16 | 6.98 |
| CIFAR100, VGG11, $T8$ | AT+EL | 69.56 | 19.68 | 11.15 | 10.13 | 20.91 |
| CIFAR100, VGG11, $T8$ | RAT+EL | 69.47 | 19.71 | 11.39 | 10.65 | 24.10 |
| CIFAR100, VGG11, $T8$ | StoG+EL | 72.58 | 8.98 | 0.58 | 0.28 | 23.54 |

Table 6: Performance (%) of the proposed evolutionary leak factor $\lambda$ (EL) with other strategies, where 'FEEL, $(\|\lambda\|_2)$' represents EL with $L_2$ norm regularization, 'GP' represents gradient penalty, which adds $L_2$ norm constraint to the model gradient. The perturbation $\epsilon = 4/255$ for all attacks, and iterative step $k = 4$, step size $\alpha = 0.01$ for PGD. The dataset is CIFAR10 with $T = 4$, the network is VGG11.

| Method | Clean | FGSM | PGD | BIM | CW |
|---|---|---|---|---|---|
| Vanilla | 92.64 | 35.47 | 15.59 | 14.95 | 6.92 |
| FEEL ($\lambda = 1.0$) or FE | 92.26 | 39.67 | 21.56 | 21.05 | 10.12 |
| FEEL ($\lambda = 0.8$) | 92.45 | 39.83 | 23.19 | 22.40 | 11.04 |
| FEEL ($\lambda = 0.5$) | 90.31 | 42.72 | 24.05 | 23.12 | 11.71 |
| FEEL ($\lambda = 0.3$) | 89.26 | 52.20 | 38.02 | 37.01 | 12.35 |
| FEEL ($\lambda = 0.0$) | 81.76 | **62.84** | **63.80** | **63.09** | 12.46 |
| FEEL, $(\|\lambda\|_2)$ | 88.52 | 44.41 | 29.98 | 29.14 | **13.89** |
| FEEL (learnable $\lambda$) or Ours | **92.73** | 44.25 | 30.27 | 29.34 | 12.39 |
| GP | 92.63 | 38.77 | 17.53 | 16.60 | 8.07 |
| GP+EL | 90.53 | 41.72 | 26.33 | 25.09 | 20.21 |
| GP+FEEL | 92.53 | **48.46** | **32.83** | **31.94** | **20.75** |

the output in this case. As can also be observed from Tab. 6, a smaller $\lambda$ could also bring performance degradation for clear inputs, *i.e.*, from 92.26% at $\lambda = 1.0$ to 81.76% at $\lambda = 0.0$. To mitigate the performance degradation, we implement the leak factor as a learnable parameter.

2). In the second case, it is difficult to directly analyze the influence of $\lambda$ on other terms in Eq. (6) due to their complex relationship. Therefore, we analyze the influence by validating whether ① term for robustness improvement affects ② term or ③ term's effectiveness for the same goal. To be specific, as analyzed in main paper, RAT (weights regularization) is essentially minimizing ② term. And we now add a gradient constraint via the L2 norm (gradient penalty regularization (GP)) to minimize ③ term. We conduct additional comparisons with these two methods to two alternatives of our methods. These two alternatives are implemented by additionally optimizing $\lambda$ for methods RAT and GP (keeping remaining parts unchanged), represented as RAT+EL and GP+EL, respectively. As shown in Tab. of the main paper and Tab. 5 and Tab. 6, RAT+EL and GP+EL significantly improve the robustness of RAT and GP, across different attack types and datasets, respectively. These results show that leveraging ① term for robustness improvement does not interfere with ② term or ③ term's effectiveness for the same goal, indicating that the leak factor does not affect other terms in Eq. (6).

In summary, results in both cases indicate that the rationality of the theoretical framework and the leak factor does not affect other terms in Eq. (6) on SNN robustness.

